# Comparative Gene Prediction using Conditional Random Fields

**Jade P. Vinson**[*][†]
jpvinson@broad.mit.edu

**David DeCaprio**[*]
daved@broad.mit.edu

**Matthew D. Pearson**
mdp@broad.mit.edu

**Stacey Luoma**
sluoma@broad.mit.edu

**James E. Galagan**
jgalag@broad.mit.edu

**The Broad Institute of MIT and Harvard**
Cambridge, MA 02142

## Abstract

Computational gene prediction using generative models has reached a plateau, with several groups converging to a generalized hidden Markov model (GHMM) incorporating phylogenetic models of nucleotide sequence evolution. Further improvements in gene calling accuracy are likely to come through new methods that incorporate additional data, both comparative and species specific. Conditional Random Fields (CRFs), which directly model the conditional probability $P(y|x)$ of a vector of hidden states conditioned on a set of observations, provide a unified framework for combining probabilistic and non-probabilistic information and have been shown to outperform HMMs on sequence labeling tasks in natural language processing.

We describe the use of CRFs for comparative gene prediction. We implement a model that encapsulates both a phylogenetic-GHMM (our baseline comparative model) and additional non-probabilistic features. We tested our model on the genome sequence of the fungal human pathogen *Cryptococcus neoformans*. Our baseline comparative model displays accuracy comparable to the the best available gene prediction tool for this organism. Moreover, we show that discriminative training and the incorporation of non-probabilistic evidence significantly improve performance.

Our software implementation, *Conrad*, is freely available with an open source license at http://www.broad.mit.edu/annotation/conrad/.

## 1 Introduction

Gene prediction is the task of labeling nucleotide sequences to identify the location and components of genes (Figure 1). The accurate automated prediction of genes is essential to both downstream bioinformatic analyses and the interpretation of biological experiments. Currently, the most accurate approach to computational gene prediction is generative modeling. In this approach, one models the joint probability of the hidden gene structure $y$ and the observed nucleotide sequence $x$. The model parameters $\theta$ are chosen to maximize the joint probability of the training data. Given a new set of observations $x$, one predicts genes by selecting the path of hidden labels $y$ that maximizes $Pr_\theta(y, x)$. Several independent groups have converged on the same generative model: a phylogenetic generalized hidden Markov model with explicit state durations (phylo-GHMM) [1, 2, 3, 4].

---

[*]These authors contributed equally
[†]Current email: jade@rentec.com

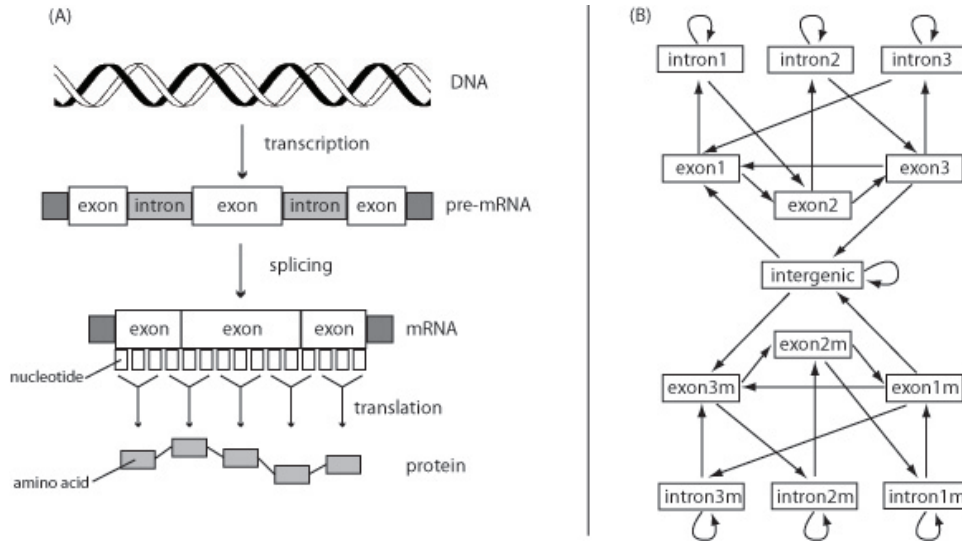

Figure 1: The processes of RNA transcription, intron splicing, and protein translation (panel A) and a state diagram for gene structure (panel B). The mirror-symmetry reflects the fact that DNA is double-stranded and genes occur on both strands. The 3-periodicity in the state diagram corresponds to the translation of nucleotide triplets into amino acids.

Further improvements in accuracy are likely to come from the incorporation of additional biological signals or new types of experimental data. However, incorporating each signal requires a handcrafted modification which increases the complexity of the generative model: a new theoretical approach is needed.

One approach to combining multiple sources of information for gene prediction, conditional maximum likelihood, was proposed in 1994 by Stormo and Haussler [5] and later implemented in the program GAZE by Howe et. al. [6, 7]. In this approach, one defines a Boltzmann distribution where the probability of each hidden sequence is proportional to the exponential of a weighted sum of different types of evidence. One then trains the weights to maximize the conditional probability $Pr_w(y|x)$ of the hidden sequence given the observations in the training data.

A related approach is the use of conditional random fields (CRFs), recently introduced in the context of natural language processing [8]. Like the earlier work in gene prediction, CRFs assign a probability to each hidden sequence that is proportional to the exponential of a weighted sum, and the weights are trained to maximize the conditional probability of the training data. The global convergence guarantee for training weights (Section 2.1 and [8]) is one of the strengths of this approach, but was not noticed in the earlier work on gene prediction. In addition, CRFs were presented in a more abstract framework and have since been applied in several domains.

Here, we apply chain-structured CRFs to gene prediction. We introduce a novel strategy for feature selection, allowing us to directly incorporate the best existing generative models with additional sources of evidence in the same theoretical framework. First, we use *probabalistic features* based on generative models whenever well-developed models are available. In this way we instantiate a phylo-GHMM as a variant of a CRF. Second, we add *non-probabilistic features* for information that is not easily modeled generatively, such as alignments of expressed sequence tags (ESTs). We developed *Conrad*, a gene predictor and highly optimized CRF engine. Conrad is freely available with an open source license at http://www.broad.mit.edu/annotation/conrad/.

We applied Conrad to predict genes in the fungal human pathogen *Cryptococcus neoformans*. Our baseline comparative model is as accurate as Twinscan [9, 10], the most accurate gene predictor trained for *C. neoformans*. Training the weights of our model discriminatively further improves prediction accuracy, indicating that discriminatively trained models can outperform generatively trained models on the same data. The addition of non-probabilistic features further improves prediction accuracy.

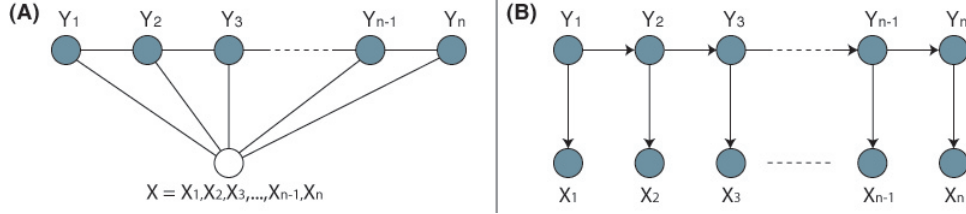

Figure 2: Graphical models for a first-order chain-structured conditional random field (panel A) and a first-order hidden Markov model (panel B). The variables $Y_i$ are hidden states and the variables $X_i$ are observations. The unshaded node is not generated by the model.

## 2  Conditional Random Fields

Conditional random fields are a framework for expressing the conditional probability $\Pr(\vec{y}|\mathbf{x})$ of hidden states $\vec{y} = (y_1, y_2, \ldots, y_n)$ given observations $\mathbf{x}$ [8]. The conditional probabilities assigned by a CRF are proportional to a weighted exponential sum of feature functions:

$$\Pr(\vec{y}|\mathbf{x}) = \frac{1}{Z_\lambda(\mathbf{x})} \exp\left( \sum_{i=1}^{n} \sum_{j \in J} \lambda_j f_j(y_{i-1}, y_i, \mathbf{x}, i) \right), \tag{1}$$

where $Z_\lambda(\mathbf{x})$ is the normalizing constant or partition function, $y_0 = \mathrm{start}$, and $J$ is the collection of features. The conditional probabilities can be viewed as a Boltzman distribution where the pairwise energy between two positions $i-1$ and $i$ is a weighted sum of the feature functions $f_j$. See Figure 2. The feature functions $f_j(y_{i-1}, y, \mathbf{x}, i)$ can be any real-valued functions defined for all possible hidden states $y_{i-1}$ and $y$, observations $\mathbf{x}$, and positions $i$. For example, the value of the feature function at position $i$ might depend on the value of the observations $\mathbf{x}$ at a distant position, allowing one to capture long-range interactions with a CRF. Varying the weights $\lambda$ of a CRF, we obtain a family of conditional probabilities $Pr_\lambda(\vec{y}|\mathbf{x})$.

An alternative viewpoint comes by reversing the order of summation in Equation 1 and expressing the conditional probability using feature sums $F_j$ that depend on the entire hidden sequence $\vec{y}$:

$$\Pr(\vec{y}|\mathbf{x}) = \frac{1}{Z_\lambda(\mathbf{x})} \exp\left( \sum_{j \in J} \lambda_j F_j(\vec{y}, \mathbf{x}) \right), \quad \text{where} \quad F_j(\vec{y}, \mathbf{x}) = \sum_{i=1}^{n} f_j(y_{i-1}, y_i, \mathbf{x}, i). \tag{2}$$

Some of the theoretical properties of CRFs, such as global convergence of weight training, can be derived using only the feature sums $F_j$. These theoretical derivations also apply to generalizations of CRFs, such as semi-Markov CRFs [11], in which one modifies the formula expressing the feature sums $F_j$ in terms of the feature functions $f_j$.

### 2.1  Inference and Training

Given a CRF and observations $\mathbf{x}$, the inference problem is to determine the sequence of hidden states with the highest conditional probability, $\vec{y}_{max} = \mathrm{argmax}_{\vec{y}}(\Pr(\vec{y}|\mathbf{x}))$. For a linear-chain CRF, each feature function $f_j$ depends only on pairs of adjacent hidden states and there is an efficient Viterbi algorithm for solving the inference problem.

Given training data $(\vec{y}, \mathbf{x})$, a CRF is trained in two steps. In the first step, free parameters associated with individual feature functions are fixed using the training data. The training methods can be specific to each feature.

In the second step, the weights $\lambda$ are selected to maximize the conditional log-likelihood:

$$\lambda_{max} \quad = \quad \underset{\lambda}{\mathrm{argmax}}\left( \log\left( Pr_\lambda(\vec{y}|\mathbf{x}) \right) \right)$$

The log-likelihood is a concave function of $\lambda$ (its Hessian is the negative covariance matrix of the random variables $F_j$ relative to the Boltzmann distribution). Thus, various iterative methods, such

as a gradient-based function optimizer[12], are guaranteed to converge to a global maximum. Using the weights obtained by training, the resulting probability distribution on $Pr_\lambda(\cdot|\mathbf{x})$ is the maximum entropy distribution subject to the constraints that the expected value of each feature sum $F_j$ is equal to its value in the training data.

One can also train the weights of the CRF to maximize an alternative objective function. For example, one can maximize the expected value $G_{AOF}(\lambda) = E_{Pr_\lambda}(S(y, y^0, \mathbf{x}))$ of the similarity between the actual hidden sequence $y^0$ and a random hidden sequence $y$ selected according to Equation 1. This objective function can be optimized using standard gradient-based function optimizers. The gradient is $\frac{\partial}{\partial \lambda_j}G_{AOF}(\lambda) = Cov_{Pr_\lambda}(S(y, y^0, \mathbf{x}), F_j(y, \mathbf{x}))$. Global convergence is not guaranteed because the objective function is not necessarily concave. When the similarity function $S$ can be defined in terms of a purely local comparison between the actual hidden sequence and a random hidden sequence, as in $S(y, y^0, \mathbf{x}) = \sum_{i=1}^{n} s(y_{i-1}, y_i, y_{i-1}^0, y_i^0, \mathbf{x}, i)$, the gradient can be efficiently computed using dynamic programming – this is the level of generality we implemented in Conrad. In this paper we consider this simplest possible alternate objective function, where the local similarity function is 1 at position i if the hidden sequence is correct and 0 otherwise. In this case the alternate objective function is just the expected number of correctly predicted positions.

## 2.2 Expressing an HMM as a CRF

Any conditional probability $\Pr(\vec{y}|\mathbf{x})$ that can be implicitly expressed using an HMM [13] can also be expressed using a CRF. Indeed, the HMM and its corresponding CRF form a generative-discriminative pair [14]. For example, a first-order HMM with transition matrix $T$, emissions matrix $B$, and initial hidden state probabilities $\vec{\pi}$ assigns the joint probability

$$Pr(\vec{y}, \vec{x}) = \pi_{y_1} \prod_{i=1}^{\text{length}-1} T_{y_i, y_{i+1}} \prod_{i=1}^{\text{length}} B_{y_i, x_i}.$$

Given an observation sequence $\mathbf{x}$, the conditional probabilities implied by this HMM can be expressed as a CRF by defining the following three features and setting all weights to $1.0$:

$$f_\pi(z, w, \mathbf{x}, i) = \begin{cases} \log(\pi_w) & \text{if } z = \text{start and } i = 1 \\ 0 & \text{otherwise} \end{cases}$$

$$f_T(z, w, \mathbf{x}, i) = \begin{cases} \log(T_{z,w}) & \text{if } i > 1 \\ 0 & \text{otherwise} \end{cases}$$

$$f_B(z, w, \mathbf{x}, i) = \log(B_{w, x_i})$$

Hidden Markov models can be extended in various directions. One of the most important extensions for gene prediction is to explicitly model state durations: for many species the lengths of some components are tightly constrained, such as introns in fungi. The extensions of HMMs to generalized HMMs (GHMMs) and CRFs to semi-Markov CRFs [11] are straightforward but omitted for clarity.

# 3 Our Model

The core issue in designing a CRF is the selection of feature functions. The approach usually taken in natural language processing is to define thousands or millions of features, each of which are indicator functions: 0 most of the time and 1 in specific circumstances. However, for gene prediction there are well-developed probabilistic models that can serve as a starting point in the design of a CRF. We propose a new approach to CRF feature selection with the following guiding principle: use probabilistic models for feature functions when possible and add non-probabistic features only where necessary. The CRF training algorithm determines the relative contributions of these features through discriminative training, without having to assume independence between the features or explicitly model dependencies.

Our approach to gene prediction is implemented as *Conrad*, a highly configurable Java executable. The CRF engine for Conrad uses LBFGS as the gradient solver for training [12, 15, 16] and is highly optimized for speed and memory usage. Because the CRF engine is a separate module with a well-defined interface for feature functions, it can also be used for applications other than gene prediction.

## 3.1 The baseline comparative model: a phylogenetic GHMM

Phylogenetic generalized hidden Markov models are now the standard approach to gene prediction using generative models [1, 2, 3, 4], and capture many of the signals for resolving gene structure (e.g. splice models or phylogenetic models of nucleotide sequence evolution). We define probabilistic features that, when taken together with weights 1.0, reproduce the phylo-GHMM that we refer to as our baseline comparative model.

Our baseline comparative model is based on the state diagram of Figure 1, enforces the basic gene constraints (e.g. open reading frames and GT-AG splice junctions), explicitly models intron length using a mixture model, and uses a set of multiply aligned genomes (including the reference genome to be annotated) as observations. The model comprises 29 feature functions of which the following are representative:

$$f_1 = \delta(y_{i-1} = \text{exon2} \ \& \ y_i = \text{intron2}) \log \left( Pr(x_{i-3} \dots x_{i+5}) \right),$$
$$\text{using a splice donor model trained by maximum likelihood.}$$
$$f_2 = \delta(y_i = \text{exon3}) \log(Pr(\text{multiple alignment column | reference nucleotide })),$$
$$\text{using a phylogenetic evolutionary model trained by ML.}$$

## 3.2 Non-probabalistic features

For many signals useful in resolving gene structure (e.g. protein homology, ESTs, CPG islands, or chromatin methylation), a probabilistic model is elusive or is difficult to incorporate in the existing framework. To explore the addition of non-probablistic evidence, we introduce two groups of feature functions, both using 0-1 indicator functions. The first group of feature functions is based on the alignment of expressed sequence tags (ESTs) to the reference genome (ESTs are the experimentally determined sequences of randomly sampled mRNA; see Figure 1):

$$f_{\text{EST},1} = \delta(y_i = \text{exon} \ \& \ \text{EST aligned at position i})$$
$$f_{\text{EST},2} = \delta(y_i = \text{intron} \ \& \ \text{position i is in the gap of an EST alignment })$$

The second group of feature functions is based on the presence of gaps in the multiple alignment, indicative of insertions or deletions (indels) in the evolutionary history of one of the aligned species. Indels are known to be relevant to gene prediction: evolution preserves the functions of most genes and an indel that is not a multiple of three would dirsupt the translation of a protein. Thus, indels not a multiple of three provide evidence against a position being part of an exon. We introduce the features

$$f_{\text{GAP},1} = \delta(y_i = \text{exon} \ \& \ \text{an indel of length 0 mod 3 has a boundary at position i })$$
$$f_{\text{GAP},2} = \delta(y_i = \text{exon} \ \& \ \text{an indel of length 1 or 2 mod 3 has a boundary at position i }),$$

plus the four analogous features for introns and intergenic regions.

For both classes of evidence, no satisfying probabilistic models exist. For example, the most systematic attempt at incorporating multiple alignment gaps in a generative model is [17], but this model only represents the case of phylogenetically simple, non-overlapping gaps.

# 4 Results

We evaluated our model using the genome of fungal human pathogen *Cryptococcus neoformans* strain JEC21 [18]. *C. neoformans* is an ideal test case due to the availability of genomes for four closely related strains for use as comparative data and a high-quality manual annotation with deep EST sequencing.

To determine an absolute benchmark, we compared our baseline comparative model to Twinscan [9, 10], the most accurate comparative gene predictor trained for *C. neoformans*. Because Twinscan was an input to the manual annotation, we evaluated the accuracy of both predictors by comparing to the alignments of ESTs (which are independent of both predictors) along an entire chromosome (chr 9). At the locations containing both an EST and a gene prediction, the accuracy of our model is comparable to (or better than) that of Twinscan. See Table 1.

Table 1: Comparing the prediction accuracy of our baseline comparative model with that of Twinscan. Accuracy statistics are collected at loci where an EST overlaps with a gene prediction.

|  | Baseline Comparative Model | Twinscan |
|---|---|---|
| Nucleotide sensitivity (%) | 99.71 | 98.35 |
| Nucleotide specificity (%) | 99.26 | 99.56 |
| Splice sensitivity (%) | 94.51 | 93.93 |
| Splice specificity (%) | 95.80 | 93.20 |

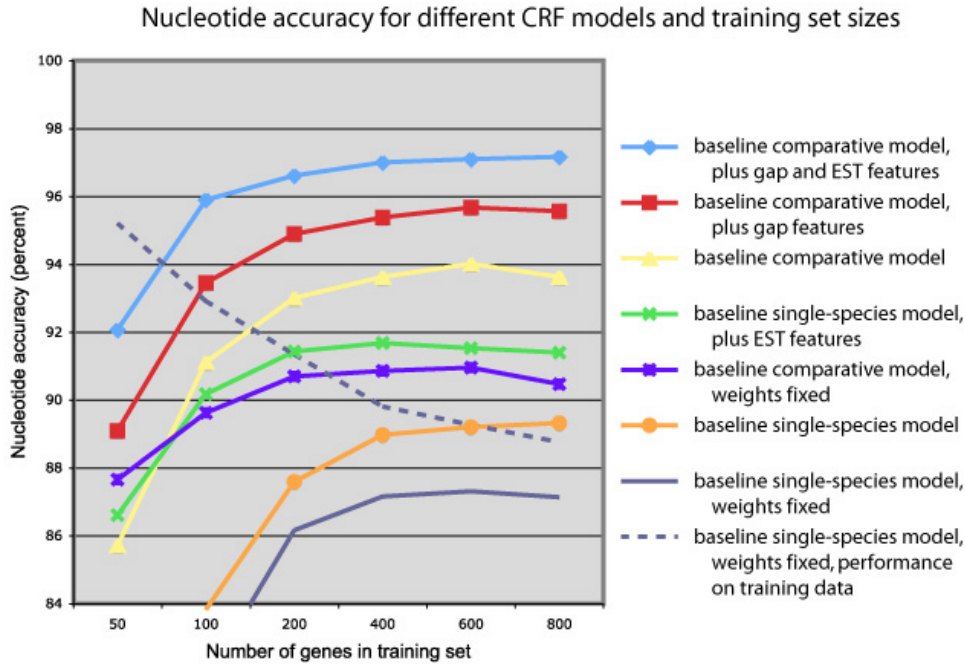

Figure 3: Gene prediction accuracy increases with additional features and with the training of feature weights. All models were trained with the alternate objective function (see text), with the exception of models labeled "weights fixed". For the latter, feature weights were fixed at 1.0. Performance on training data (dashed line), performance on testing data (solid lines). Each data point above is the average of 10 cross-validation replicates.

We next measured the relative effects of different sets of features and methods for training the feature weights. First, we created a set of 1190 trusted genes by selecting those genes which had EST support along their entire length. We then performed 10 cross-validation replicates for several combinations of a set of features and a method for training weights, and a training set sizes (50, 100, 200, 400, or 800 genes). For each set of replicates, we record the average nucleotide accuracy. See Figure 3. As expected, the testing accuracy increases with larger training sets, while the training accuracy decreases. Note that for these experiments, we do not explicitly model intron length.

## 4.1   Adding features improves accuracy

The effect of including additional features is shown in Figure 3. As can be seen in each case, model accuracy improves as new evidence is added. For a 400 gene training set, adding the EST features increases the accuracy of the baseline single species model from 89.0% to 91.7%. Adding the gap features increases the accuracy of the baseline comparative model from 93.6% to 95.4%. Finally, adding both types of evidence together increases accuracy more than either addition in isolation: adding EST and gap features to the baseline comparative model increases accuracy from 93.6% to 97.0%. Ongoing work is focused on including many additional lines of evidence.

## 4.2 Training using an alternate objective function

The standard training of weights for CRFs seeks to maximize the conditional log probability of the training data. However, this approach has limitations: one would like to use an objective function that is closely related to evaluation criteria relevant to the problem domain. Previous work in natural language processing found no accuracy benefit to changing the objective function [19]. However, relative to the usual training to maximize conditional log-likelihood, we observed about 2% greater nucleotide accuracy in testing data using models trained to maximize an alternative objective function (the expected nucleotide accuracy of a random sequence on training data). See Section 2.1.

The results shown in Figure 3 are all using this alternate objective function. For example, for a 400 gene training set, training the weights increases the accuracy of the baseline single species model from 87.2% to 89% and the baseline comparative model from 90.9% to 93.6%.

## 5 Concluding Remarks

CRFs are a promising framework for gene prediction. CRFs offer several advantages relative to standard HMM-based gene prediction methods including the ability to capture long-range dependencies and to incorporate heterogeneous data within a single framework. We have implemented a semi-Markov CRF by explicitly expressing a phylogenetic GHMM within a CRF framework and extending this baseline with non-probabilisitic evidence. When used to predict genes in the fungal human pathogen *C. neoformans*, our model displays accuracy comparable to the best existing gene prediction tools. Moreover, we show that incorporation of non-probabilistic evidence improves performance.

The key issue in designing CRFs is the selection of feature functions, and our approach differs from previous applications. We adopt the following guiding principle: we use probabilistic models as features where possible and incorporate non-probabilistic features only when necessary. In contrast, in natural language processing features are typically indicator functions. Our approach also differs from an initial study of using CRFs for gene prediction [20], which does not use a probabilistic model as the baseline.

CRFs offer a solution to an important problem in gene prediction: how to combine probabilistic models of nucleotide sequences with additional evidence from diverse sources. Prior research in this direction has focused on either handcrafted heuristics for a particular type of feature [21], a mixture-of-experts approach applied at each nucleotide position [22], and decision trees [23]. CRFs offer an alternative approach in which probabilistic features and non-probabilistic evidence are both incorporated in the same framework.

CRFs are applicable to other discriminative problems in bioinformatics. For example, CRFs can be used train optimal parameters for protein sequence alignment [24]. In all these examples, as with gene predictions, CRFs provide the ability to incorporate supplementary evidence not captured in current generative models.

## Acknowledgement

This work has been supported by NSF grant number MCB-0450812. We thank Nick Patterson for frequent discussions on generative probabilistic modeling. We thank Richard Durbin for recognizing the connection to the earlier work by Stormo and Haussler. We thank the anonymous reviews for indicating which aspects of our work warranted more or less detail relative to the initial submission.

## References

[1] Adam Siepel and David Haussler. Combining phylogenetic and hidden Markov models in biosequence analysis. *J Comput Biol*, 11(2-3):413–428, 2004.

[2] Jon D McAuliffe, Lior Pachter, and Michael I Jordan. Multiple-sequence functional annotation and the generalized hidden Markov phylogeny. *Bioinformatics*, 20(12):1850–1860, Aug 2004.

[3] Jakob Skou Pedersen and Jotun Hein. Gene finding with a hidden Markov model of genome structure and evolution. *Bioinformatics*, 19(2):219–227, Jan 2003.

[4] Randall H Brown, Samuel S Gross, and Michael R Brent. Begin at the beginning: predicting genes with 5' UTRs. *Genome Res*, 15(5):742–747, May 2005.

[5] G. D. Stormo and D. Haussler. Optimally parsing a sequence into different classes based on multiple types of information. In *Proc. of Second Int. Conf. on Intelligent Systems for Molecular Biology*, pages 369–375, Menlo Park, CA, 1994. AAAI/MIT Press.

[6] Kevin L Howe, Tom Chothia, and Richard Durbin. GAZE: a generic framework for the integration of gene-prediction data by dynamic programming. *Genome Res*, 12(9):1418–1427, Sep 2002.

[7] Kevin L. Howe. *Gene prediction using a configurable system for the integration of data by dynamic programming*. PhD thesis, University of Cambridge, 2003.

[8] John Lafferty, Andrew McCallum, and Fernando Pereira. Conditional random fields: Probabilistic models for segmenting and labeling sequence data. In *Proc. 18th International Conf. on Machine Learning*, pages 282–289. Morgan Kaufmann, San Francisco, CA, 2001.

[9] Aaron E Tenney, Randall H Brown, Charles Vaske, Jennifer K Lodge, Tamara L Doering, and Michael R Brent. Gene prediction and verification in a compact genome with numerous small introns. *Genome Res*, 14(11):2330–2335, Nov 2004.

[10] I Korf, P Flicek, D Duan, and M R Brent. Integrating genomic homology into gene structure prediction. *Bioinformatics*, 17 Suppl 1:140–148, 2001.

[11] S. Sarawagi and W. Cohen. Semimarkov conditional random fields for information extraction. *Proceedings of ICML*, 2004.

[12] Richard H. Byrd, Peihuang Lu, Jorge Nocedal, and Ci You Zhu. A limited memory algorithm for bound constrained optimization. *SIAM Journal on Scientific Computing*, 16(6):1190–1208, 1995.

[13] Lawrence Rabiner. A tutorial on hidden markov models and selected applications in speech recognition. In Alex Waibel and Kai-Fu Lee, editors, *Readings in speech recognition*, pages 267–296. Morgan Kaufmann, San Mateo, 1990.

[14] Charles Sutton and Andrew McCallum. An introduction to conditional random fields for relational learning. In Lise Getoor and Ben Taskar, editors, *Statistical Relational Learning*. To appear.

[15] Hanna Wallach. Efficient training of conditional random fields. Master's thesis, University of Edinburgh, 2002.

[16] F. Sha and F. Pereira. Shallow parsing with conditional random fields. Technical Report CIS TR MS-CIS-02-35, University of Pennsylvania, 2003.

[17] Adam Siepel and David Haussler. Computational identification of evolutionarily conserved exons. In *Proceedings of the 8th Annual International Conference, RECOMB 2004*. ACM, 2004.

[18] Brendan J Loftus and Eula Fung et. al. The genome of the basidiomycetous yeast and human pathogen Cryptococcus neoformans. *Science*, 307(5713):1321–1324, Feb 2005.

[19] Yasemin Altun, Mark Johnson, and Thomas Hofmann. Investigating Loss Functions and Optimization Methods for Discriminative Learning of Label Sequences. *Proceedings of the 2003 Conference on Empirical Methods in Natural Language Processing*.

[20] Aron Culotta, David Kulp, and Andrew McCallum. Gene prediction with conditional random fields. Technical Report UM-CS-2005-028, University of Massachusetts, Amherst, April 2005.

[21] R F Yeh, L P Lim, and C B Burge. Computational inference of homologous gene structures in the human genome. *Genome Res*, 11(5):803–816, May 2001.

[22] Brona Brejova, Daniel G Brown, Ming Li, and Tomas Vinar. ExonHunter: a comprehensive approach to gene finding. *Bioinformatics*, 21 Suppl 1:i57–i65, Jun 2005.

[23] Jonathan E Allen and Steven L Salzberg. JIGSAW: integration of multiple sources of evidence for gene prediction. *Bioinformatics*, 21(18):3596–3603, Sep 2005.

[24] Chuong B. Do, Samuel S. Gross, and Serafim Batzoglou. Contralign: Discriminative training for protein sequence alignment. In Alberto Apostolico, Concettina Guerra, Sorin Istrail, Pavel A. Pevzner, and Michael S. Waterman, editors, *RECOMB*, volume 3909 of *Lecture Notes in Computer Science*, pages 160–174. Springer, 2006.
